# High Performance Neural Net Simulation on a Multiprocessor System with "Intelligent" Communication

Urs A. Müller, Michael Kocheisen, and Anton Gunzinger
Electronics Laboratory, Swiss Federal Institute of Technology
CH-8092 Zurich, Switzerland

## Abstract

The performance requirements in experimental research on artificial neural nets often exceed the capability of workstations and PCs by a great amount. But speed is not the only requirement. Flexibility and implementation time for new algorithms are usually of equal importance. This paper describes the simulation of neural nets on the MUSIC parallel supercomputer, a system that shows a good balance between the three issues and therefore made many research projects possible that were unthinkable before. (MUSIC stands for Multiprocessor System with Intelligent Communication)

## 1  Overview of the MUSIC System

The goal of the MUSIC project was to build a fast parallel system and to use it in real-world applications like neural net simulations, image processing or simulations in chemistry and physics [1, 2]. The system should be flexible, simple to program and the realization time should be short enough to not have an obsolete system by the time it is finished. Therefore, the fastest available standard components were used. The key idea of the architecture is to support the collection and redistribution of complete data blocks by a simple, efficient and autonomously working communication network realized in hardware. Instead of considering where to send data and where from to receive data, each processing element determines which part of a (virtual) data block it has produced and which other part of the same data block it wants to receive for the continuation of the algorithm.

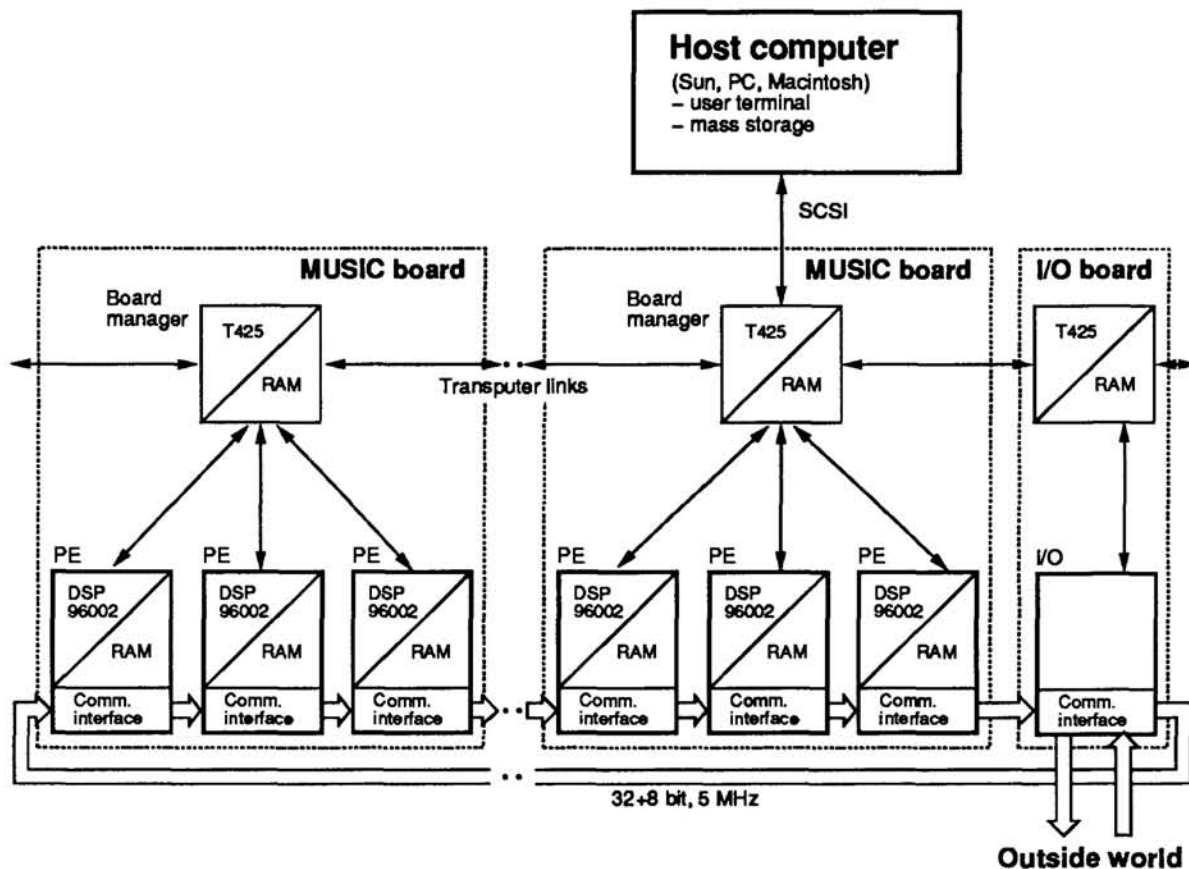

Figure 1: Overview of the MUSIC hardware

Figure 1 shows an overview of the MUSIC architecture. For the realization of the communication paradigm a ring architecture has been chosen. Each processing element has a communication interface realized with a XILINX 3090 programmable gate array. During communication the data is shifted through a 40-bit wide bus (32 bit data and 8 bit token) operated at a 5-MHz clock rate. On each clock cycle, the processing elements shift a data value to their right neighbors and receive a new value from their left neighbors. By counting the clock cycles each communication interface knows when to copy data from the stream passing by into the local memory of its processing element and, likewise, when to insert data from the local memory into the ring. The tokens are used to label invalid data and to determine when a data value has circulated through the complete ring.

Three processing elements are placed on a 9 × 8.5-inch board, each of them consisting of a Motorola 96002 floating-point processor, 2 Mbyte video (dynamic) RAM, 1 Mbyte static RAM and the above mentioned communication controller. The video RAM has a parallel port which is connected to the processor and a serial port which is connected to the communication interface. Therefore, data processing is almost not affected by the communication network's activity and communication and processing can overlap in time. This allows to use the available communication bandwidth more efficiently. The processors run at 40 MHz with a peak performance of 60 MFlops. Each board further contains an Inmos T425 transputer as a board

| | |
|---|---|
| Number of processing elements: | 60 |
| Peak performance: | 3.6 Gflops |
| Floating-point format: | 44 bit IEEE single extended precision |
| Memory: | 180 Mbyte |
| Programming language: | C, Assembler |
| Cabinet: | 19-inch rack |
| Cooling: | forced air cooling |
| Total power consumption: | less than 800 Watt |
| Host computer: | Sun workstation, PC or Macintosh |

Table 1: MUSIC system technical data

manager, responsible for performance measurements and data communication with the host (a Sun workstation, PC or Macintosh).

In order to provide the fast data throughput required by many applications, special I/O modules (for instance for real-time video processing applications) can be added which have direct access to the fast ring bus. An SCSI interface module for four parallel SCSI-2 disks, which is currently being developed, will allow the storage of huge amount of training data for neural nets. Up to 20 boards (60 processing elements) fit into a standard 19-inch rack resulting in a 3.6-Gflops system. MUSIC's technical data is summarized in Table 1.

For programming the communication network just three library functions are necessary: Init_comm() to specify the data block dimensions and data partitioning, Data_ready() to label a certain amount of data as ready for communication and Wait_data() to wait for the arrival of the expected data (synchronization). Other functions allow the exchange and automatic distribution of data blocks between the host computer and MUSIC and the calling of individual user functions. The activity of the transputers is embedded in these functions and remains invisible for the user.

Each processing element has its own local program memory which makes MUSIC a MIMD machine (multiple instructions multiple data). However, there is usually only one program running on all processing elements (SPMD = single program multiple data) which makes programming as simple or even simpler as programming a SIMD computer (single instruction multiple data). The difference to SIMD machines is that each processor can take different program pathes on conditional branches without the performance degradation that occurs on SIMD computers in such a case. This is especially important for the simulation of neural nets with nonregular local structures.

## 2    Parallelization of Neural Net Algorithms

The first implemented learning algorithm on MUSIC was the well-known back-propagation applied to fully connected multilayer perceptrons [3]. The motivation was to gain experience in programming the system and to demonstrate its performance on a real-world application. All processing elements work on the same layer a time, each of them producing an individual part of the output vector (or error vector in the backward path) [1]. The weights are distributed to the processing elements accordingly. Since a processing element needs different weight subsets in

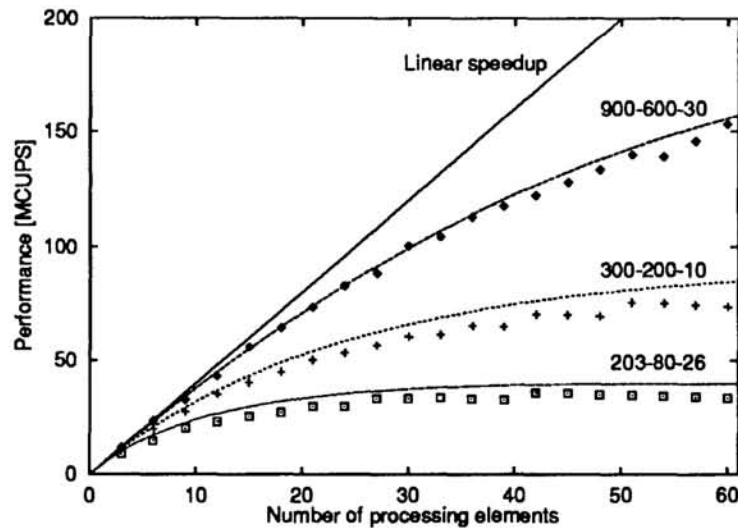

Figure 2: Estimated (lines) and measured (points) back-propagation performance for different neural net sizes.

the forward and in the backward path, two subsets are stored and updated on each processing element. Each weight is therefore stored and updated twice on different locations on the MUSIC system [1]. This is done to avoid the communication of the weights during learning what would cause a saturation of the communication network. The estimated and experimentally measured speedup for different sizes of neural nets is illustrated in Figure 2.

Another frequently reported parallelization scheme is to replicate the complete network on all processing elments and to let each of them work on an individual subset of the training patterns [4, 5, 6]. The implementation is simpler and the communication is reduced. However, it does not allow continuous weight update, which is known to converge significantly faster than batch learning in many cases. A comparison of MUSIC with other back-propagation implementations reported in the literature is shown in Table 2.

Another category of neural nets that have been implemented on MUSIC are cellular neural nets (CNNs) [10]. A CNN is a two-dimensional array of nonlinear dynamic cells, where each cell is only connected to a local neighborhood [11, 12]. In the MUSIC implementation every processing elment computes a different part of the array. Between iteration steps only the overlapping parts of the neighborhoods need to be communicated. Thus, the computation to communication ratio is very high resulting in an almost linear speedup up to the maximum system size. CNNs are used in image processing and for the modeling of biological structures.

## 3    A Neural Net Simulation Environment

After programming all necessary functions for a certain algorithm (e.g. forward propagate, backward propagate, weight update, etc.) they need to be combined

| System | No. of PEs | Performance | | | Cont. weight update |
| | | Forward [MCPS] | Learning [MCUPS] | Peak (%) | |
|---|---|---|---|---|---|
| PC (80486, 50 MHz)* | 1 | 1.1 | 0.47 | 38.0 | Yes |
| Sun (Sparcstation 10)* | 1 | 3.0 | 1.1 | 43.0 | Yes |
| Alpha Station (150 MHz)* | 1 | 8.3 | 3.2 | 8.6 | Yes |
| Hypercluster [7] | 64 | 27.0 | 9.9 | — | — |
| Warp [4] | 10 | — | 17.0 | — | No |
| CM-2** [6] | 64K | 180.0 | 40.0 | — | No |
| Cray Y-MP C90*** | 1 | 220.3 | 65.6 | — | Yes |
| RAP [8] | 40 | 574.0 | 106.0 | 50.0 | Yes |
| NEC SX-3*** | 1 | — | 130.0 | 9.6 | Yes |
| MUSIC* | 60 | 504.0 | 247.0 | 28.0 | Yes |
| Sandy/8** [9] | 256 | — | 583.0 | 31.0 | Yes |
| GF11 [5] | 356 | — | 901.0 | 54.0 | No |

*Own measurements
**Estimated numbers
***No published reference available.

Table 2: Comparison of floating-point back-propagation implementations. "PEs" means processing elements, "MCPS" stands for millions of connections per second in the forward path and "MCUPS" is the number of connection updates per second in the learning mode, including both forward and backward path. Note that not all implementations allow continuous weight update.

in order to construct and train a specific neural net or to carry out a series of experiments. This can be done using the same programming language that was used to program the neural functions (in case of MUSIC this would be C). In this case the programmer has maximum flexibility but he also needs a good knowledge of the system and programming language and after each change in the experimental setup a recompilation of the program is necessary.

Because a set of neural functions is usually used by many different researchers who, in many cases, don't want to be involved in a low-level (parallel) programming of the system, it is desirable to have a simpler front-end for the simulator. Such a front-end can be a shell program which allows to specify various parameters of the algorithm (e.g. number of layers, number of neurons per layer, etc.). The usage of such a shell can be very easy and changes in the experimental setup don't require recompilation of the code. However, the flexibility for experimental research is usually too much limited with a simple shell program. We have chosen a way in between: a command language to combine the neural functions which is interactive and much simpler to learn and to use than an ordinary programming language like C or Fortran. The command language should have the following properties:

- interactive
- easy to learn and to use
- flexible
- loops and conditional branches
- variables
- transparent interface to neural functions.

Instead of defining a new special purpose command language we decided to consider an existing one. The choice was Basic which seems to meet the above requirements best. It is easy to learn and to use, it is widely spread, flexible and interactive. For this purpose a Basic interpreter, named *Neuro-Basic*, was written that allows the calling of neural (or other) functions running parallel on MUSIC. From the Basic level itself the parallelism is completely invisible. To allocate a new layer with 300 neurons, for instance, one can type

```
a = new_layer(300)
```

The variable *a* afterwards holds a pointer to the created layer which later can be used in other functions to reference that layer. The following command propagates layer *a* to layer *b* using the weight set *w*

```
propagate(a, b, w)
```

Other functions allow the randomization of weights, the loading of patterns and weight sets, the computation of mean squared errors and so on. Each instruction can be assigned to a program line and can then be run as a program. The sequence

```
10 a = new_layer(300)
20 b = new_layer(10)
30 w = new_weights(a, b)
```

for instance defines a two-layer perceptron with 300 input and 10 output neurons being connected with the weights *w*. Larger programs, loops and conditional branches can be used to construct and train complete neural nets or to automatically run complete series of experiments where experimental setups depend on the result of previous experiments. The Basic environment thus allows all kinds of gradations in experimental research, from the interactive programming of small experiments till large off-line learning jobs. Extending the simulator with new learning algorithms means that the programmer just has to write the parallel code of the actual algorithm. It can then be controlled by a Basic program and it can be combined with already existing algorithms.

The Basic interpreter runs on the host computer allowing easy access to the input/output devices of the host. However, the time needed for interpreting the commands on the host can easily be in the same order of magnitude as the runtime of the actual functions on the attached parallel processor array. The interpretation of a Basic program furthermore is a sequential part of the system (it doesn't run faster if the system size is increased) which is known to be a fundamental limit in speedup (Amdahls law [13]). Therefore the Basic code is not directly interpreted on the host but first is compiled to a simpler stack oriented meta-code, named b-code, which is afterwards copied and run on all processing elements at optimum speed. The compilation phase is not really noticeable to the user since compiling 1000 source lines takes less than a second on a workstation.

Note that Basic is not the programming language for the MUSIC system, it is a high level command language for the easy control of parallel algorithms. The actual programming language for MUSIC is C or Assembler.

Of course, Neuro-Basic is not restricted to the MUSIC system. The same principle can be used for neural net simulation on conventional workstations, vector computers or other parallel systems. Furthermore, the parallel algorithms of MUSIC also run on sequential computers. Simulations in Neuro-Basic can therefore be executed locally on a workstation or PC as well.

## 4   Conclusions

Neuro-Basic running on MUSIC proved to be an important tool to support experimental research on neural nets. It made possible to run many experiments which could not have been carried out otherwise. An important question, however, is, how much more programming effort is needed to implement a new algorithm in the Neuro-Basic environment compared to an implementation on a conventional workstation and how much faster does it run.

| Algorithm | additional programming | speedup |
|---|---|---|
| Back-propagation (C) | × 2 | 60 |
| Back-propagation (Assembler) | × 8 | 240 |
| Cellular neural nets (CNN) | × 3 | 60 |

Table 3: Implementation time and performance ratio of a 60-processor MUSIC system compared to a Sun Sparcstation-10

Table 3 contains these numbers for back-propagation and cellular neural nets. It shows that if an additional programming effort of a factor two to three is invested to program the MUSIC system in C, the return of investment is a speedup of approximately 60 compared to a Sun Sparcstation-10. This means one year of CPU time on a workstation corresponds to less than a week on the MUSIC system.

**Acknowledgements**

We would like to express our gratitude to the many persons who made valuable contributions to the project, especially to Peter Kohler and Bernhard Bäumle for their support of the MUSIC system, José Osuna for the CNN implementation and the students Ivo Hasler, Björn Tiemann, René Hauck, Rolf Krähenbühl who worked for the project during their graduate work.

This work was funded by the Swiss Federal Institute of Technology, the Swiss National Science Foundation and the Swiss Commission for Support of Scientific Research (KWF).

## References

[1] Urs A. Müller, Bernhard Bäumle, Peter Kohler, Anton Gunzinger, and Walter Guggenbühl. Achieving supercomputer performance for neural net simulation with an array of digital signal processors. *IEEE Micro Magazine*, 12(5):55–65, October 1992.

[2] Anton Gunzinger, Urs A. Müller, Walter Scott, Bernhard Bäumle, Peter Kohler, Hansruedi Vonder Mühll, Florian Müller-Plathe, Wilfried F. van Gunsteren, and Walter Guggenbühl. Achieving super computer performance with a DSP array processor. In Robert Werner, editor, *Supercomputing '92*, pages 543–550. IEEE/ACM, IEEE Computer Society Press, November 16–20, 1992, Minneapolis, Minnesota 1992.

[3] D. E. Rumelhart, G. E. Hinton, and R. J. Williams. Learning internal representation by error propagation. In David E. Rumelhart and James L. McClelland, editors, *Parallel Distributet Processing: Explorations in the Microstructure of Cognition*, volume 1, pages 318–362. Bradford Books, Cambridge MA, 1986.

[4] Dean A. Pomerleau, George L. Gusciora, David S. Touretzky, and H. T. Kung. Neural network simulation at Warp speed: How we got 17 million connections per second. In *IEEE International Conference on Neural Networks*, pages II.143–150, July 24–27, San Diego, California 1988.

[5] Michael Witbrock and Marco Zagha. An implementation of backpropagation learning on GF11, a large SIMD parallel computer. *Parallel Computing*, 14(3):329–346, 1990.

[6] Xiru Zhang, Michael Mckenna, Jill P. Mesirov, and David L. Waltz. An efficient implementation of the back-propagation algorithm on the Connection Machine CM-2. In David S. Touretzky, editor, *Advances in Neural Information Processing Systems (NIPS-89)*, pages 801–809, 2929 Campus Drive, Suite 260, San Mateo, CA 94403, 1990. Morgan Kaufmann Publishers.

[7] Heinz Mühlbein and Klaus Wolf. Neural network simulation on parallel computers. In David J. Evans, Gerhard R. Joubert, and Frans J. Peters, editors, *Parallel Computing-89*, pages 365–374, Amsterdam, 1990. North Holland.

[8] Phil Kohn, Jeff Bilmes, Nelson Morgan, and James Beck. Software for ANN training on a Ring Array Processor. In John E. Moody, Steven J. Hanson, and Richard P. Lippmann, editors, *Advances in Neural Information Processing Systems 4 (NIPS-91)*, 2929 Campus Drive, Suite 260, San Mateo, California 94403, 1992. Morgan kaufmann.

[9] Hideki Yoshizawa, Hideki Kato Hiroki Ichiki, and Kazuo Asakawa. A highly parallel architecture for back-propagation using a ring-register data path. In *2nd International Conference on Microelectrnics for Neural Networks (ICMNN-91)*, pages 325–332, October 16–18, Munich 1991.

[10] J. A. Osuna, G. S. Moschytz, and T. Roska. A framework for the classification of auditory signals with cellular neural networks. In H. Dedieux, editor, *Procedings of 11. European Conference on Circuit Theory and Design*, pages 51–56 (part 1). Elsevier, August 20 – Sept. 3 Davos 1993.

[11] Leon O. Chua and Lin Yang. Cellular neural networks: Theory. *IEEE Transactions on Circuits and Systems*, 35(10):1257–1272, October 1988.

[12] Leon O. Chua and Lin Yang. Cellular neural networks: Applications. *IEEE Transactions on Circuits and Systems*, 35(10):1273–1290, October 1988.

[13] Gene M. Amdahl. Validity of the single processor approach to achieving large scale computing capabilities. In *AFIPS Spring Computer Conference Atlantic City, NJ*, pages 483–485, April 1967.